# Neural Reconstruction with Approximate Message Passing (NeuRAMP)

**Alyson K. Fletcher**
University of California, Berkeley
alyson@eecs.berkeley.edu

**Sundeep Rangan**
Polytechnic Institute of New York University
srangan@poly.edu

**Lav R. Varshney**
IBM Thomas J. Watson Research Center
lrvarshn@us.ibm.com

**Aniruddha Bhargava**
University of Wisconsin Madison
aniruddha@wisc.edu

## Abstract

Many functional descriptions of spiking neurons assume a cascade structure where inputs are passed through an initial linear filtering stage that produces a low-dimensional signal that drives subsequent nonlinear stages. This paper presents a novel and systematic parameter estimation procedure for such models and applies the method to two neural estimation problems: (i) compressed-sensing based neural mapping from multi-neuron excitation, and (ii) estimation of neural receptive fields in sensory neurons. The proposed estimation algorithm models the neurons via a graphical model and then estimates the parameters in the model using a recently-developed generalized approximate message passing (GAMP) method. The GAMP method is based on Gaussian approximations of loopy belief propagation. In the neural connectivity problem, the GAMP-based method is shown to be computational efficient, provides a more exact modeling of the sparsity, can incorporate nonlinearities in the output and significantly outperforms previous compressed-sensing methods. For the receptive field estimation, the GAMP method can also exploit inherent structured sparsity in the linear weights. The method is validated on estimation of linear nonlinear Poisson (LNP) cascade models for receptive fields of salamander retinal ganglion cells.

## 1  Introduction

Fundamental to describing the behavior of neurons in response to sensory stimuli or to inputs from other neurons is the need for succinct models that can be estimated and validated with limited data. Towards this end, many functional models assume a cascade structure where an initial linear stage combines inputs to produce a low-dimensional output for subsequent nonlinear stages. For example, in the widely-used linear nonlinear Poisson (LNP) model for retinal ganglion cells (RGCs) [1,2], the time-varying input stimulus vector is first linearly filtered and summed to produce a low (typically one or two) dimensional output, which is then passed through a memoryless nonlinear function that outputs the neuron's instantaneous Poisson spike rate. An initial linear filtering stage also appears in the well-known integrate-and-fire model [3]. The linear filtering stage in these models reduces the dimensionality of the parameter estimation problem and provides a simple characterization of a neuron's receptive field or connectivity.

However, even with the dimensionality reduction from assuming such linear stages, parameter estimation may be difficult when the stimulus is high-dimensional or the filter lengths are large. Compressed sensing methods have been recently proposed [4] to reduce the dimensionality further. The key insight is that although most experiments for mapping, say visual receptive fields, expose the

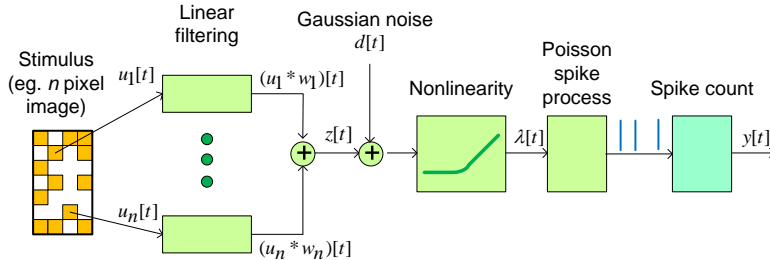

Figure 1: Linear nonlinear Poisson (LNP) model for a neuron with $n$ stimuli.

neural system under investigation to a large number of stimulus components, the overwhelming majority of the components do not affect the instantaneous spiking rate of any one particular neuron due to anatomical sparsity [5, 6]. As a result, the linear weights that model the response to these stimulus components will be sparse; most of the coefficients will be zero. For the retina, the stimulus is typically a large image, whereas the receptive field of any individual neuron is usually only a small portion of that image. Similarly, for mapping cortical connectivity to determine the connectome, each neuron is typically only connected to a small fraction of the neurons under test [7]. Due to the sparsity of the weights, estimation can be performed via sparse reconstruction techniques similar to those used in compressed sensing (CS) [8–10].

This paper presents a CS-based estimation of linear neuronal weights via a recently-developed generalized approximate message passing (GAMP) methods from [11] and [12]. GAMP, which builds upon earlier work in [13, 14], is a Gaussian approximation of loopy belief propagation. The benefits of the GAMP method for neural mapping are that it is computationally tractable with large sums of data, can incorporate very general graphical model descriptions of the neuron and provides a method for simultaneously estimating the parameters in the linear and nonlinear stages. In contrast, methods such as the common spike-triggered average (STA) perform separate estimation of the linear and nonlinear components. Following the simulation methodology in [4], we show that the GAMP method offers significantly improved reconstruction of cortical wiring diagrams over other state-of-the-art CS techniques.

We also validate the GAMP-based sparse estimation methodology in the problem of fitting LNP models of salamander RGCs. LNP models have been widely-used in systems modeling of the retina, and they have provided insights into how ganglion cells communicate to the lateral geniculate nucleus, and further upstream to the visual cortex [15]. Such understanding has also helped clarify the computational purpose of cell connectivity in the retina. The filter shapes estimated by the GAMP algorithm agree with other findings on RGC cells using STA methods, such as [16]. What is important here is that the filter coefficients can be estimated accurately with a much smaller number of measurements. This feature suggests that GAMP-based sparse modeling may be useful in the future for other neurons and more complex models.

## 2 Linear Nonlinear Poisson Model

### 2.1 Mathematical Model

We consider the following simple LNP model for the spiking output of a single neuron under $n$ stimulus components shown in Fig. 1, cf. [1, 2]. Inputs and outputs are measured in uniform time intervals $t = 0, 1, \ldots, T - 1$, and we let $u_j[t]$ denote the $j$th stimulus input in the $t$th time interval, $j = 1, \ldots, n$. For example, if the stimulus is a sequence of images, $n$ would be the number of pixels in each image and $u_j[t]$ would be the value of the $j$th pixel over time. We let $y[t]$ denote the number of spikes in the $t$th time interval, and the general problem is to find a model that explains the relation between the stimuli $u_j[t]$ and spike outputs $y[t]$.

As the name suggests, the LNP model is a cascade of three stages: linear, nonlinear and Poisson. In the first (linear) stage, the input stimulus is passed through a set of $n$ linear filters and then summed

to produce the scalar output $z[t]$ given by

$$z[t] = \sum_{j=1}^{n}(w_j * u_j)[t] = \sum_{j=1}^{n}\sum_{\ell=0}^{L-1} w_j[\ell]u_j[t-\ell], \tag{1}$$

where $w_j[\cdot]$ is the linear filter applied to the $j$th stimulus component and $(w_j * u_j)[t]$ is the convolution of the filter with the input. We assume the filters have finite impulse response (FIR) with $L$ taps, $w_j[\ell]$, $\ell = 0, 1, \ldots, L-1$. In the second (nonlinear) stage of the LNP model, the scalar linear output $z[t]$ passes through a memoryless nonlinear random function to produce a spike rate $\lambda[t]$. We assume a nonlinear mapping of the form

$$\lambda[t] \quad = \quad f(v[t]) = \log\Big[1 + \exp\big(\phi(v[t];\boldsymbol{\alpha})\big)\Big], \tag{2a}$$

$$v[t] \quad = \quad z[t] + d[t], \;\; d[t] \sim \mathcal{N}(0,\sigma_d^2), \tag{2b}$$

where $d[t]$ is Gaussian noise to account for randomness in the spike rate and $\phi(v;\boldsymbol{\alpha})$ is the $\nu$-th order polynomial,

$$\phi(v;\boldsymbol{\alpha}) = \alpha_0 + \alpha_1 v + \cdots + \alpha_\nu v^\nu. \tag{3}$$

The form of the function in (2b) ensures that the spike rate $\lambda[t]$ is always positive. In the third and final stage of the LNP model, the number of spikes is modeled as a Poisson process with mean $\lambda[t]$. That is,

$$\Pr\Big(y[t] = k \,\Big|\, \lambda[t]\Big) = e^{-\lambda[t]}\lambda[t]^k/k!, \quad k = 0, 1, 2, \ldots \tag{4}$$

This LNP model is sometimes called a *one-dimensional* model since $z[t]$ is a scalar.

## 2.2 Conventional Estimation Methods

The parameters in the neural model can be written as the vector $\boldsymbol{\theta} = (\mathbf{w}, \boldsymbol{\alpha}, \sigma_d^2)$, where $\mathbf{w}$ is the $nL$-dimensional vector of the filter coefficients, the vector $\boldsymbol{\alpha}$ contains the $\nu + 1$ polynomial coefficients in (3) and $\sigma_d^2$ is the noise variance. The basic problem is to estimate the parameters $\boldsymbol{\theta}$ from the input/output data $u_j[t]$ and $y[t]$. We briefly summarize three conventional methods: spike-triggered average (STA), reverse correlation (RC) and maximum likelihood (ML), all described in several texts including [1].

The STA and RC methods are based on simple linear regression. The vector $\mathbf{z}$ of linear filter outputs $z[t]$ in (1) can be written as $\mathbf{z} = \mathbf{A}\mathbf{w}$, where $\mathbf{A}$ is a known block Toeplitz matrix with the input data $u_j[t]$. The STA and RC methods then both attempt to find a $\mathbf{w}$ such that output $\mathbf{z}$ has high linear correlation with measured spikes $\mathbf{y}$. The RC method finds this solution with the least squares estimate

$$\widehat{\mathbf{w}}_{\mathrm{RC}} = (\mathbf{A}^*\mathbf{A} + \sigma^2 I)^{-1}\mathbf{A}^*\mathbf{y}, \tag{5}$$

for some parameter $\sigma^2$, and the STA is an approximation given by

$$\widehat{\mathbf{w}}_{\mathrm{STA}} = \frac{1}{T}\mathbf{A}^*\mathbf{y}. \tag{6}$$

The statistical properties of the estimates are discussed in [17, 18].

Once the estimate $\widehat{\mathbf{w}} = \widehat{\mathbf{w}}_{\mathrm{STA}}$ or $\widehat{\mathbf{w}}_{\mathrm{RC}}$ has been computed, one can compute an estimate $\widehat{\mathbf{z}} = \mathbf{A}\widehat{\mathbf{w}}$ for the linear output $\mathbf{z}$ and then use any scalar estimation method to find a nonlinear mapping from $z[t]$ to $\lambda[t]$ based on the outputs $y[t]$.

A shortcoming of the STA and RC methods is that the filter coefficients $\mathbf{w}$ are selected to maximize the linear correlation and may not work well when there is a strong nonlinearity. A maximum likelihood (ML) estimate may overcome this problem by jointly optimizing over nonlinear and linear parameters. To describe the ML estimate, first fix parameters $\boldsymbol{\alpha}$ and $\sigma_d^2$ in the nonlinear stage. Then, given the vector output $\mathbf{z}$ from the linear stage, the spike count components $y[t]$ are independent:

$$\Pr\Big(\mathbf{y}\,\Big|\,\mathbf{z}, \boldsymbol{\alpha}, \sigma_d^2\Big) = \prod_{t=0}^{T-1}\Pr\Big(y[t]\,\Big|\,z[t], \boldsymbol{\alpha}, \sigma_d^2\Big) \tag{7}$$

where the component distributions are given by

$$P\Big(y[t]\,\Big|\,z[t],\boldsymbol{\alpha},\sigma_d^2\Big)=\int_0^\infty \Pr\Big(y[t]\,\Big|\,\lambda[t]\Big)p\big(\lambda[t]\,\big|\,z[t],\boldsymbol{\alpha},\sigma_d^2\big)\,d\lambda[t], \tag{8}$$

and $p\big(\lambda[t]\,\big|\,z[t],\boldsymbol{\alpha},\sigma_d^2\big)$ can be computed from the relation (2b) and $\Pr\big(y[t]\,\big|\,\lambda[t]\big)$ is the Poisson distribution (4). The ML estimate is then given by the solution to the optimization

$$\widehat{\boldsymbol{\theta}}_{\mathrm{ML}}:=\underset{(\mathbf{w},\boldsymbol{\alpha},\sigma_d^2)}{\arg\max}\prod_{t=0}^{T-1}\Pr\Big(y[t]\,\Big|\,z[t],\boldsymbol{\alpha},\sigma_d^2\Big),\quad \mathbf{z}=\mathbf{Aw}. \tag{9}$$

In this way, the ML estimate attempts to maximize the goodness of fit by simultaneously searching over the linear and nonlinear parameters.

## 3    Estimation via Compressed Sensing

### 3.1    Bayesian Model with Group Sparsity

A difficulty in the above methods is that the number, $Ln$, of filter coefficients in $\mathbf{w}$ may be large and require an excessive number of measurements to estimate accurately. As discussed above, the key idea in this work is that most stimulus components have little effect on the spiking output. Most of the filter coefficients $w_j[\ell]$ will be zero and exploiting this sparsity may be able to reduce the number of measurements while maintaining the same estimation accuracy.

The sparse nature of the filter coefficients can be modeled with the following *group sparsity* structure: Let $\xi_j$ be a binary random variable with $\xi_j=1$ when stimulus $j$ is in the receptive field of the neuron and $\xi_j=0$ when it is not. We call the variables $\xi_j$ the *receptive field indicators*, and model these indicators as i.i.d. Bernoulli variables with

$$\Pr(\xi_j=1)=1-\Pr(\xi_j=0)=\rho, \tag{10}$$

where $\rho\in[0,1]$ is the average fraction of stimuli in the receptive field. We then assume that, given the vector $\boldsymbol{\xi}$ of receptive field indicators, the filter weight coefficients are independent with distribution

$$p\Big(w_j[\ell]\,\Big|\,\boldsymbol{\xi}\Big)=p\Big(w_j[\ell]\,\Big|\,\xi_j\Big)=\left\{\begin{array}{ll}0 & \text{if }\xi_j=0\\ \mathcal{N}(0,\sigma_x^2) & \text{if }\xi_j=1.\end{array}\right. \tag{11}$$

That is, the linear weight coefficients are zero outside the receptive field and Gaussian within the receptive field. Since our algorithms are general, other distributions can also be used—we use the Gaussian for illustration. The distribution on $\mathbf{w}$ defined by (10) and (11) is often called a *group sparse* model, since the components of the vector $\mathbf{w}$ are zero in groups.

Estimation with this sparse structure leads naturally to a compressed sensing problem. Specifically, we are estimating a sparse vector $\mathbf{w}$ through a noisy version $\mathbf{y}$ of a linear transform $\mathbf{z}=\mathbf{Aw}$, which is precisely the problem of compressed sensing [8–10]. With a group structure, one can employ a variety of methods including the group Lasso [19–21] and group orthogonal matching pursuit [22]. However, these methods are designed for either AWGN or logistic outputs. In the neural model, the spike count $y[t]$ is a nonlinear, random function of the linear output $z[t]$ described by the probability distribution in (8).

### 3.2    GAMP-Based Sparse Estimation

To address the nonlinearities in the outputs, we use the generalized approximate message passing (GAMP) algorithm [11] with extensions in [12]. The GAMP algorithm is a general approximate inference method for graphical models with *linear mixing*. To place the neural estimation problem in the GAMP framework, first fix the stimulus input vector $\mathbf{u}$, nonlinear output parameters $\boldsymbol{\alpha}$ and $\sigma_d^2$. Then, the conditional joint distribution of the outputs $\mathbf{y}$, linear filter weights $\mathbf{w}$ and receptive field indicators $\boldsymbol{\xi}$ factor as

$$p\Big(\mathbf{y},\boldsymbol{\xi},\mathbf{w}\,\Big|\,\mathbf{u},\boldsymbol{\alpha},\sigma_d^2\Big)\;=\;\prod_{j=1}^n\left[\Pr(\xi_j)\prod_{\ell=0}^{L-1}p(w_j[\ell]\,|\,\xi_j)\right]\prod_{t=0}^{T-1}\Pr\Big(y[t]\,\Big|\,z[t],\boldsymbol{\alpha},\sigma_d^2\Big),$$

$$\mathbf{z}\;=\;\mathbf{Aw}. \tag{12}$$

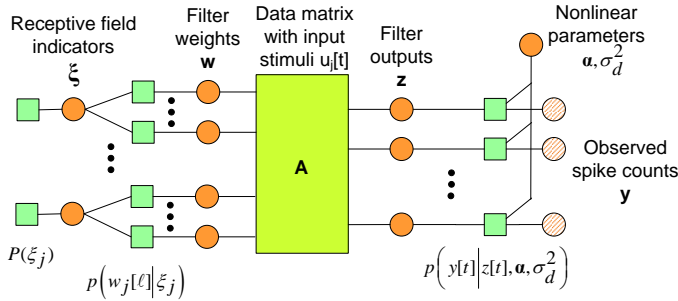

Figure 2: The neural estimation problem represented as a graphical model with linear mixing. Solid circles are unknown variables, dashed circles are observed variables (in this case, spike counts) and squares are factors in the probability distribution. The linear mixing component of the graph indicates the constraints that $\mathbf{z} = \mathbf{A}\mathbf{w}$.

Similar to standard graphical model estimation [23], GAMP is based on the first representing the distribution in (12) via a *factor graph* as shown in Fig. 2. In the factor graph, the solid circles represent the components of the unknown vectors $\mathbf{w}$, $\boldsymbol{\xi}$, ..., and the dashed circles the components of the observed or measured variables $\mathbf{y}$. Each square corresponds to one factor in the distribution (12). What is new for the GAMP methodology, is that the factor graph also contains a component to indicate the linear constraints that $\mathbf{z} = \mathbf{A}\mathbf{w}$, which would normally be represented by a set of additional factor nodes.

Inference on graphical models is often performed by some variant of loopy belief propagation (BP). Loopy BP attempts to reduce the joint estimation of all the variables to a sequence of lower dimensional estimation problems associated with each of the factors in the graph. Estimation at the factor nodes is performed iteratively, where after each iteration, "beliefs" of the variables are passed to the factors to improve the estimates in the subsequent iterations. Details can be found in [23].

However, exact implementation of loopy BP is intractable for the neural estimation problem: The linear constraints $\mathbf{z} = \mathbf{A}\mathbf{w}$ create factor nodes that connect each of the variables $z[t]$ to all the variables $w_j[\ell]$ where $u_j[t - \ell]$ is non-zero. In the RGC experiments below, the pixels value $u_j[t]$ are non-zero 50% of the time, so each variable $z[t]$ will be connected to, on average, half of the $Ln$ filter weight coefficients through these factor nodes. Since exact implementation of loopy BP grows exponentially in the degree of the factor nodes, loopy BP would be infeasible for the neural problem, even for moderate values of $Ln$.

The GAMP method reduces the complexity of loopy BP by exploiting the linear nature of the relations between the variables $\mathbf{w}$ and $\mathbf{z}$. Specifically, it is shown that when each term $z[t]$ is a linear combination of a large number of terms $w_j[\ell]$, the belief messages across the factor node for the linear constraints can be approximated as Gaussians and the factor node updates can be computed with a central limit theorem approximation. Details are in [11] and [12].

## 4   Receptive Fields of Salamander Retinal Ganglion Cells

The sparse LNP model with GAMP-based estimation was evaluated on data from recordings of neural spike trains from salamander retinal ganglion cells exposed to random checkerboard images, following the basic methods of [24].[1] In the experiment, spikes from individual neurons were measured over an approximately 1900s period at a sampling interval of 10ms. During the recordings, the salamander was exposed to $80 \times 60$ pixel random black-white binary images that changed every 3 to 4 sampling intervals. The pixels of each image were i.i.d. with a 50-50 black-white probability.

We compared three methods for fitting an $L = 30$ tap one-dimensional LNP model for the RGC neural responses: (i) truncated STA, (ii) approximate ML, and (iii) GAMP estimation with the sparse LNP model. Methods (i) and (ii) do not exploit sparsity, while method (iii) does.

The truncated STA method was performed by first computing a linear filter estimate as in (6) for the entire $80 \times 60$ image and then setting all coefficients outside an $11 \times 11$ pixel subarea around the pixel with the largest estimated response to zero. The $11 \times 11$ size was chosen since it is sufficiently large to contain these neurons' entire receptive fields. This truncation significantly improves the STA estimate by removing spurious estimates that anatomically cannot have relation to the neural

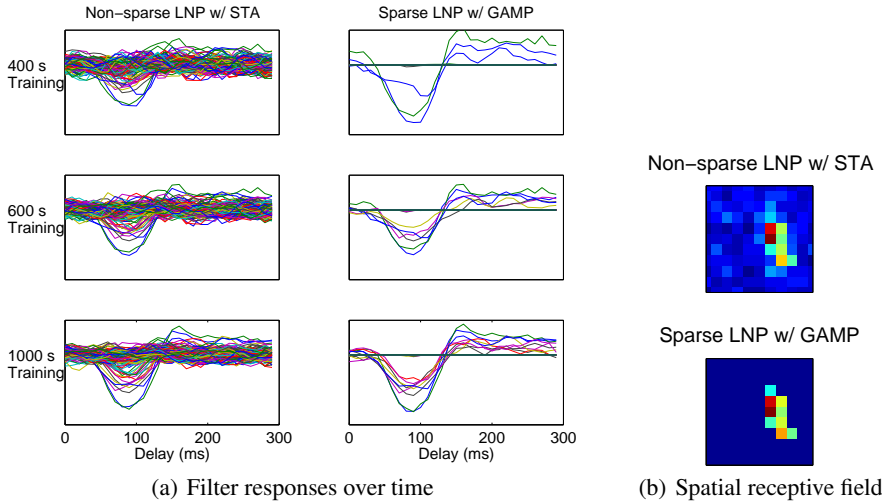

(a) Filter responses over time

(b) Spatial receptive field

Figure 3: Estimated filter responses and visual receptive field for salamander RGCs using a non-sparse LNP model with STA estimation and a sparse LNP model with GAMP estimation.

responses; this provides a better comparison to test other methods. From the estimate $\widehat{\mathbf{w}}_{\mathrm{STA}}$ of the linear filter coefficients, we compute an estimate $\widehat{\mathbf{z}} = \mathbf{A}\widehat{\mathbf{w}}$ of the linear filter output. The output parameters $\boldsymbol{\alpha}$ and $\sigma_d^2$ are then fit by numerical maximization of likelihood $P(\mathbf{y}|\widehat{\mathbf{z}}, \boldsymbol{\alpha}, \sigma_d^2)$ in (7).

We used a ($\nu = 1$)-order polynomial, since higher orders did not improve the prediction. The fact that only a linear polynomial was needed in the output is likely due to the fact that random checkerboard images rarely align with the neuron's filters and therefore do not excite the neural spiking into a nonlinear regime. An interesting future experiment would be to re-run the estimation with swatches of natural images as in [25]. We believe that under such experimental conditions, the advantages of the GAMP-based nonlinear estimation would be even larger.

The RC estimate (5) was also computed, but showed no appreciable difference from the STA estimate for this matrix $\mathbf{A}$. As a result, we discuss only STA results below.

The GAMP-based sparse estimation used the STA estimate for initialization to select the $11 \times 11$ pixel subarea and the variances $\sigma_x^2$ in (11). As in the STA case, we used only a ($\nu = 1$)-order linear polynomial in (3). The linear coefficient $\alpha_1$ was set to 1 since other scalings could be absorbed into the filter weights $\mathbf{w}$. The constant term $\alpha_0$ was incorporated as another linear regression coefficient.

For a third algorithm, we approximately computed the ML estimate (9) by running the GAMP algorithm, but with all the factors for the priors on the weights $\mathbf{w}$ removed.

To illustrate the qualitative differences between the estimates, Fig. 3 shows the estimated responses for the STA and GAMP-based sparse LNP estimates for one neuron using three different lengths of training data: 400, 600 and 1000 seconds of the total 1900 second training data. For brevity, the approximate ML estimate is omitted, but is similar to the STA estimate. The estimated responses in Fig. 3(a) are displayed as $11 \times 11 = 121$ curves, each curve representing the linear filter response with $L = 30$ taps over the $30 \times 10 = 300$ms response. Fig. 3(b) shows the estimated spatial receptive fields plotted as the total magnitude of the $11 \times 11$ filters. One can immediately see that the GAMP based sparse estimate is significantly less noisy than the STA estimate, as the smaller, unreliable responses are zeroed out in the GAMP-based sparse LNP estimate.

The improved accuracy of the GAMP-estimation with the sparse LNP model was verified in the cross validation, as shown in Fig. 4. In this plot, the length of the training data was varied from 200 to 1000 seconds, with the remaining portion of the 1900 second data used for cross-validation. At each training length, each of the three methods—STA, GAMP-based sparse LNP and approximate ML—were used to produce an estimate $\widehat{\boldsymbol{\theta}} = (\widehat{\mathbf{w}}, \widehat{\boldsymbol{\alpha}}, \widehat{\sigma}_d^2)$. Fig. 4 shows, for each of these methods, the cross-validation scores $P(\mathbf{y}|\widehat{\mathbf{z}}, \widehat{\boldsymbol{\alpha}}, \widehat{\sigma}_d^2)^{1/T}$, which is the geometric mean of the likelihood in (7). It can be seen that the GAMP-based sparse LNP estimate significantly outperforms the STA and

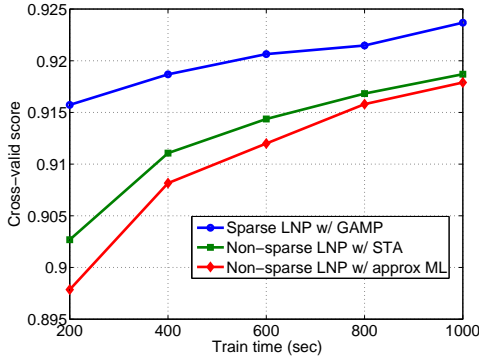

Figure 4: Prediction accuracy of sparse and non-sparse LNP estimates for data from sala-mander RGC cells. Based on cross-validation scores, the GAMP-based sparse LNP estimation provides a significantly better estimate for the same amount of training.

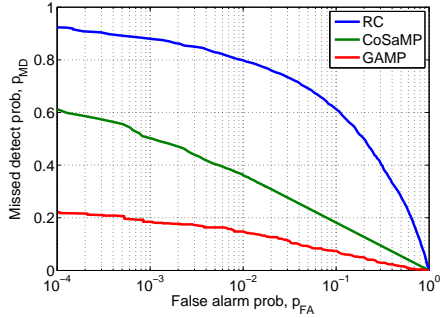

Figure 5: Comparison of reconstruction methods on cortical connectome mapping with multi-neuron excitation based on simulation model in [4]. In this case, connectivity from $n = 500$ potential pre-synaptic neurons are estimated from $m = 300$ measurements with 40 neurons excited in each measurement. In the simulation, only 6% of the $n$ potential neurons are actually connected to the post-synaptic neuron under test.

approximate ML estimates that do not assume any sparse structure. Indeed, by the measure of the cross-validation score, the sparse LNP estimate with GAMP after only 400 seconds of data was as accurate as the STA estimate with 1000 seconds of data. Interestingly, the approximate ML estimate is actually worse than the STA estimate, presumably since it overfits the model.

## 5   Neural Mapping via Multi-Neuron Excitation

The GAMP methodology was also applied to neural mapping from multi-neuron excitation, orig-inally proposed in [4]. A single post-synaptic neuron has connections to $n$ potential pre-synaptic neurons. The standard method to determine which of the $n$ neurons are connected to the post-synaptic neurons is to excite one neuron at a time. This process is wasteful, since only a small fraction of the neurons are typically connected. In the method of [4], multiple neurons are excited in each measurement. Then, exploiting the sparsity in the connectivity, compressed sensing tech-niques can be used to recover the mapping from $m < n$ measurements. Unfortunately, the output stage of spiking neurons is often nonlinear and most CS methods cannot directly incorporate such nonlinearities into the estimation. The GAMP methodology thus offers the possibility of improved performance for reconstruction.

To validate the methodology, we compared the performance of GAMP to various reconstruction methods following a simulation of mapping of cortical neurons with multi-neuron excitation in [4]. The simulation assumes an LNP model of Section 2.1, where the inputs $u_j[t]$ are 1 or 0 depending on whether the $j$th pre-synaptic input is excited in $t$th measurement. The filters have a single tap (i.e. $L$=1), which are modeled as a Bernoulli-Weibull distribution with a probability $\rho = 0.06$ of being on (the neuron is connected) or $1 - \rho$ of being zero (the neuron is not connected). The output has a strong nonlinearity including a thresholding and saturation – the levels of which must be estimated. Connectivity detection amounts to determining which of the $n$ pre-synaptic neurons have non-zero weights.

Fig. 5 plots the missed detection vs. false alarm rate of the various detectors. It can be seen that the GAMP-based connectivity detection significantly outperforms both non-sparse RC reconstruction as well as a state-of-the-art greedy sparse method CoSaMP [26, 27].

## 6  Conclusions and Future Work

A general method for parameter estimation in neural models based on generalized approximate message passing was presented. The GAMP methodology is computationally tractable for large data sets, can exploit sparsity in the linear coefficients and can incorporate a wide range of nonlinear modeling complexities in a systematic manner. Experimental validation of the GAMP-based estimation of a sparse LNP model for salamander RGC cells shows significantly improved prediction in cross-validation over simple non-sparse estimation methods such as STA. Benefits over state-of-the-art sparse reconstruction methods are also apparent in simulated models of cortical mapping with multi-neuron excitation.

Going forward, the generality offered by the GAMP model will enable accurate parameter estimation for other complex neural models. For example, the GAMP model can incorporate other prior information such as a correlation between responses in neighboring pixels. Future work may also include experiments with integrate-and-fire models [3]. An exciting future possibility for cortical mapping is to decode memories, which are thought to be stored as the connectome [7, 28].

Throughout this paper, we have presented GAMP as an experimental data analysis method. One might wonder, however, whether the brain itself might use compressive representations and message-passing algorithms to make sense of the world. There have been several previous suggestions that visual and general cortical regions of the brain may use belief propagation-like algorithms [29, 30]. There have also been recent suggestions that the visual system uses compressive representations [31]. As such, we assert the biological plausibility of the brain itself using the algorithms presented herein for receptive field and memory decoding.

## 7  Acknowledgements

We thank D. B. Chklovskii and T. Hu for formulative discussions on the problem, A. Leonardo for providing experimental data and further discussions, and B. Olshausen for discussions.

## Footnotes

[1]Data from the Leonardo Laboratory at the Janelia Farm Research Campus.

## References

[1] Peter Dayan and L. F. Abbott. *Theoretical Neuroscience. Computational and Mathematical Modeling of Neural Systems*. MIT Press, 2001.

[2] Odelia Schwartz, Jonathan W. Pillow, Nicole C. Rust, and Eero P. Simoncelli. Spike-triggered neural characterization. *J. Vis.*, 6(4):13, July 2006.

[3] Liam Paninski, Jonathan W. Pillow, and Eero P. Simoncelli. Maximum Likelihood Estimation of a Stochastic Integrate-and-Fire Neural Encoding Model. *Neural Computation*, 16(12):2533–2561, December 2004.

[4] Tao Hu and Dmitri B. Chklovskii. Reconstruction of sparse circuits using multi-neuronal excitation (RESCUME). In Yoshua Bengio, Dale Schuurmans, John Lafferty, Chris Williams, and Aron Culotta, editors, *Advances in Neural Information Processing Systems 22*, pages 790–798. MIT Press, Cambridge, MA, 2009.

[5] James R. Anderson, Bryan W. Jones, Carl B. Watt, Margaret V. Shaw, Jia-Hui Yang, David DeMill, James S. Lauritzen, Yanhua Lin, Kevin D. Rapp, David Mastronarde, Pavel Koshevoy, Bradley Grimm, Tolga Tasdizen, Ross Whitaker, and Robert E. Marc. Exploring the retinal connectome. *Mol. Vis*, 17:355–379, February 2011.

[6] Elad Ganmor, Ronen Segev, and Elad Schneidman. The architecture of functional interaction networks in the retina. *J. Neurosci.*, 31(8):3044–3054, February 2011.

[7] Lav R. Varshney, Per Jesper Sjöström, and Dmitri B. Chklovskii. Optimal information storage in noisy synapses under resource constraints. *Neuron*, 52(3):409–423, November 2006.

[8] E. J. Candès, J. Romberg, and T. Tao. Robust uncertainty principles: Exact signal reconstruction from highly incomplete frequency information. *IEEE Trans. Inform. Theory*, 52(2):489–509, February 2006.

[9] D. L. Donoho. Compressed sensing. *IEEE Trans. Inform. Theory*, 52(4):1289–1306, April 2006.

[10] E. J. Candès and T. Tao. Near-optimal signal recovery from random projections: Universal encoding strategies? *IEEE Trans. Inform. Theory*, 52(12):5406–5425, December 2006.

[11] S. Rangan. Generalized Approximate Message Passing for Estimation with Random Linear Mixing. arXiv:1010.5141 [cs.IT]., October 2010.

[12] S. Rangan, A.K. Fletcher, V.K.Goyal, and P. Schniter. Hybrid Approximate Message Passing with Applications to Group Sparsity . arXiv, 2011.

[13] D. Guo and C.-C. Wang. Random sparse linear systems observed via arbitrary channels: A decoupling principle. In *Proc. IEEE Int. Symp. Inform. Th.*, pages 946 – 950, Nice, France, June 2007.

[14] David L. Donoho, Arian Maleki, and Andrea Montanari. Message-passing algorithms for compressed sensing. *PNAS*, 106(45):18914–18919, September 2009.

[15] David H. Hubel. *Eye, Brain, and Vision*. W. H. Freeman, 2nd edition, 1995.

[16] Toshihiko Hosoya, Stephen A. Baccus, and Markus Meister. Dynamic predictive coding by the retina. *Nature*, 436(7047):71–77, July 2005.

[17] E. J. Chichilnisky. A simple white noise analysis of neuronal light responses. *Network: Computation in Neural Systems.*, 12:199–213, 2001.

[18] L. Paninski. Convergence properties of some spike-triggered analysis techniques. *Network: Computation in Neural Systems*, 14:437–464, 2003.

[19] S. Bakin. *Adaptive regression and model selection in data mining problems*. PhD thesis, Australian National University, Canberra, 1999.

[20] M. Yuan and Y. Lin. Model selection and estimation in regression with grouped variables. *J. Royal Statist. Soc.*, 68:49–67, 2006.

[21] Lukas Meier, Sara van de Geer, and Peter Bühlmann. Model selection and estimation in regression with grouped variables. *J. Royal Statist. Soc.*, 70:53–71, 2008.

[22] Aurélie C. Lozano, Grzegorz Świrszcz, and Naoki Abe. Group orthogonal matching pursuit for variable selection and prediction. In *Proc. NIPS*, Vancouver, Canada, December 2008.

[23] C. M. Bishop. *Pattern Recognition and Machine Learning*. Information Science and Statistics. Springer, New York, NY, 2006.

[24] Markus Meister, Jerome Pine, and Denis A. Baylor. Multi-neuronal signals from the retina: acquisition and analysis. *J. Neurosci. Methods*, 51(1):95–106, January 1994.

[25] Joaquin Rapela, Jerry M. Mendel, and Norberto M. Grzywacz. Estimating nonlinear receptive fields from natural images. *J. Vis.*, 6(4):11, May 2006.

[26] D. Needell and J. A. Tropp. CoSaMP: Iterative signal recovery from incomplete and inaccurate samples. *Appl. Comput. Harm. Anal.*, 26(3):301–321, May 2009.

[27] W. Dai and O. Milenkovic. Subspace pursuit for compressive sensing signal reconstruction. *IEEE Trans. Inform. Theory*, 55(5):2230–2249, May 2009.

[28] Dmitri B. Chklovskii, Bartlett W. Mel, and Karel Svoboda. Cortical rewiring and information storage. *Nature*, 431(7010):782–788, October 2004.

[29] Tai Sing Lee and David Mumford. Hierarchical bayesian inference in the visual cortex. *J. Opt. Soc. Am. A*, 20(7):1434–1448, July 2003.

[30] Karl Friston. The free-energy principle: a unified brain theory? *Nat. Rev. Neurosci.*, 11(2):127–138, February 2010.

[31] Guy Isely, Christopher J. Hillar, and Friedrich T. Sommer. Decyphering subsampled data: Adaptive compressive sampling as a principle of brain communication. In J. Lafferty, C. K. I. Williams, J. Shawe-Taylor, R. S. Zemel, and A. Culotta, editors, *Advances in Neural Information Processing Systems 23*, pages 910–918. MIT Press, Cambridge, MA, 2010.

